# Near-Uniform Sampling of Combinatorial Spaces Using XOR Constraints

**Carla P. Gomes**    **Ashish Sabharwal**    **Bart Selman**
Department of Computer Science
Cornell University, Ithaca NY 14853-7501, USA
{gomes,sabhar,selman}@cs.cornell.edu *

## Abstract

We propose a new technique for sampling the solutions of combinatorial problems in a near-uniform manner. We focus on problems specified as a Boolean formula, i.e., on SAT instances. Sampling for SAT problems has been shown to have interesting connections with probabilistic reasoning, making practical sampling algorithms for SAT highly desirable. The best current approaches are based on Markov Chain Monte Carlo methods, which have some practical limitations. Our approach exploits combinatorial properties of random parity (XOR) constraints to prune away solutions near-uniformly. The final sample is identified amongst the remaining ones using a state-of-the-art SAT solver. The resulting sampling distribution is provably arbitrarily close to uniform. Our experiments show that our technique achieves a significantly better sampling quality than the best alternative.

## 1    Introduction

We present a new method, XORSample, for uniformly sampling from the solutions of hard combinatorial problems. Although our method is quite general, we focus on problems expressed in the Boolean Satisfiability (SAT) framework. Our work is motivated by the fact that efficient sampling for SAT can open up a range of interesting applications in probabilistic reasoning [6, 7, 8, 9, 10, 11]. There has also been a growing interest in combining logical and probabilistic constraints as in the work of Koller, Russell, Domingos, Bacchus, Halpern, Darwiche, and many others (see e.g. statistical relational learning and Markov logic networks [1]), and a recently proposed Markov logic system for this task uses efficient SAT sampling as its core reasoning mechanism [2].

Typical approaches for sampling from combinatorial spaces are based on Markov Chain Monte Carlo (MCMC) methods, such as the Metropolis algorithm and simulated annealing [3, 4, 5]. These methods construct a Markov chain with a predefined stationary distribution. One can draw samples from the stationary distribution by running the Markov chain for a sufficiently long time. Unfortunately, on many combinatorial problems, the time taken by the Markov chain to reach its stationary distribution scales exponentially with the problem size.

MCMC methods can also be used to find (globally optimal) solutions to combinatorial problems. For example, simulated annealing (SA) uses the Boltzmann distribution as the stationary distribution. By lowering the temperature parameter to near zero, the distribution becomes highly concentrated around the minimum energy states, which correspond to the solutions of the combinatorial problem under consideration. SA has been successfully applied to a number of combinatorial search problems. However, many combinatorial problems, especially those with intricate constraint structure, are beyond the reach of SA and related MCMC methods. Not only does problem structure make reaching the stationary distribution prohibitively long, even reaching a single (optimal) solution is often infeasible. Alternative combinatorial search techniques have been developed that are much more effective at finding solutions. These methods generally exploit clever search space pruning

techniques, which quickly focus the search on small, but promising, parts of the overall combinatorial space. As a consequence, these techniques tend to be highly biased, and sample the set of solutions in an extremely non-uniform way. (Many are in fact deterministic and will only return one particular solution.)

In this paper, we introduce a general probabilistic technique for obtaining near-uniform samples from the set of all (globally optimal) solutions of combinatorial problems. Our method can use any state-of-the-art specialized combinatorial solver as a subroutine, without requiring any modifications to the solver. The solver can even be deterministic. Most importantly, the quality of our sampling method is not affected by the possible bias of the underlying specialized solver — all we need is a solver that is good at finding *some* solution or proving that none exists. We provide theoretical guarantees for the sampling quality of our approach. We also demonstrate the practical feasibility of our approach by sampling near-uniformly from instances of hard combinatorial problems.

As mentioned earlier, to make our discussion more concrete, we will discuss our method in the context of SAT. In the SAT problem, we have a set of logical constraints on a set of Boolean (True/False) variables. The challenge is to find a setting of the variables such that all logical constraints are satisfied. SAT is the prototypical NP-complete problem, and quite likely the most widely studied combinatorial problem in computer science. There have been dramatic advances in recent years in the state-of-the-art of SAT solvers [e.g. 12, 13, 14]. Current solvers are able to solve problems with millions of variables and constraints. Many practical combinatorial problems can be effectively translated into SAT. As a consequence, one of the current most successful approaches to solving hard computational problems, arising in, *e.g.*, hardware and software verification and planning and scheduling, is to first translate the problem into SAT, and then use a state-of-the-art SAT solver to find a solution (or show that it does not exist). As stated above, these specialized solvers derive much of their power from quickly focusing their search on a very small part of the combinatorial space. Many SAT solvers are deterministic, but even when the solvers incorporate some randomization, solutions will be sampled in a highly non-uniform manner.

The central idea behind our approach can be summarized as follows. Assume for simplicity that our original SAT instance on $n$ Boolean variables has $2^s$ solutions or satisfying assignments. How can we sample uniformly at random from the set of solutions? We add special randomly generated logical constraints to our SAT problem. Each random constraint is constructed in such a way that it rules out any given truth assignment exactly with probability $1/2$. Therefore, in expectation, after adding $s$ such constraints, we will have a SAT instance with exactly one solution.[1] We then use a SAT solver to find the remaining satisfying assignment and output this as our first sample. We can repeat this process with a new set of $s$ randomly generated constraints and in this way obtain another random solution. Note that to output each sample, we can use whatever off-the-shelf SAT solver is available, because all it needs to do is find the single remaining assignment.[2] The randomization in the added constraints will guarantee that the assignment is selected uniformly at random.

How do we implement this approach? For our added constraints, we use randomly generated parity or "exclusive-or" (XOR) constraints. In recent work, we introduced XOR constraints for the problem of counting the number of solutions using `MBound` [15]. Although the building blocks of `MBound` and `XORSample` are the same, this work relies much more heavily on the properties of XOR constraints, namely, pairwise and even 3-wise independence. As we will discuss below, an XOR constraint eliminates any given truth assignment with probability $1/2$, and therefore, in expectation, cuts the set of satisfying assignments in half. For this expected behavior to happen often, the elimination of each assignment should ideally be fully independent of the elimination of other assignments. Unfortunately, as far as is known, there are no compact (polynomial size) logical constraints that can achieve such complete independence. However, XOR constraints guarantee at least pairwise independence, i.e., if we know that an XOR constraint $C$ eliminates assignment $\sigma_1$, this provides no information as to whether $C$ will remove any another assignment $\sigma_2$. Remarkably, as we will see, such pairwise independence already leads to near-uniform sampling.

Our sampling approach is inspired by earlier work in computational complexity theory by Valiant and Vazirani [16], who considered the question whether having one or more assignments affects

the hardness of combinatorial problems. They showed that, in essence, the number of solutions should not affect the hardness of the problem instances in the worst case [16]. This was received as a negative result because it shows that finding a solution to a Unique SAT problem (a SAT instance that is guaranteed to have at most one solution) is not any easier than finding a solution to an arbitrary SAT instance. Our sampling strategy turns this line of research into a positive direction by showing how a standard SAT solver, tailored to finding just one solution of a SAT problem, can now be used to sample near-uniformly from the set of solutions of an arbitrary SAT problem.

In addition to introducing XORSample and deriving theoretical guarantees on the quality of the samples it generates, we also provide an empirical validation of our approach. One question that arises is whether the state-of-the-art SAT solvers will perform well on problem instances with added XOR (or parity) constraints. Fortunately, as our experiments show, a careful addition of such constraints does generally not degrade the performance of the solvers. In fact, the addition of XOR constraints can be beneficial since the constraints lead to additional propagation that can be exploited by the solvers.[3] Our experiments show that we can effectively sample near-uniformly from hard practical combinatorial problems. In comparison with the best current alternative method on such instances, our sampling quality is substantially better.

## 2 Preliminaries

For the rest of this paper, fix the set of propositional variables in all formulas to be $V$, $|V| = n$. A *variable assignment* $\sigma : V \rightarrow \{0,1\}$ is a function that assigns a value in $\{0,1\}$ to each variable in $V$. We may think of the value 0 as FALSE and the value 1 as TRUE. We will often abuse notation and write $\sigma(i)$ for valuations of entities $i \notin V$ when the intended meaning is either already defined or is clear from the context. In particular, $\sigma(1) = 1$ and $\sigma(0) = 0$. When $\sigma(i) = 1$, we say that $\sigma$ *satisfies* $i$. For $x \in V$, $\neg x$ denotes the corresponding *negated* variable; $\sigma(\neg x) = 1 - \sigma(x)$. Let $F$ be a formula over variables $V$. $\sigma(F)$ denotes the valuation of $F$ under $\sigma$. If $\sigma$ satisfies $F$, i.e., $\sigma(F) = 1$, then $\sigma$ is a *model*, *solution*, or *satisfying assignment* for $F$. Our goal in this paper is to *sample uniformly* from the set of all solutions of a given formula $F$.

An XOR *constraint* $D$ over variables $V$ is the logical "xor" or parity of a subset of $V \cup \{1\}$; $\sigma$ satisfies $D$ if it satisfies an *odd number* of elements in $D$. The value 1 allows us to express even parity. For instance, $D = \{a,b,c,1\}$ represents the xor constraint $a \oplus b \oplus c \oplus 1$, which is TRUE when an even number of $a,b,c$ are TRUE. Note that it suffices to use only positive variables. E.g., $\neg a \oplus b \oplus \neg c$ and $\neg a \oplus b$ are equivalent to $D = \{a,b,c\}$ and $D = \{a,b,1\}$, respectively. Our focus will be on formulas which are a logical conjunction of a formula in Conjunctive Normal Form (CNF) and some XOR constraints. In all our experiments, XOR constraints are translated into CNF using additional variables so that the full formula can be fed directly to standard (CNF-based) SAT solvers.

We will need basic concepts from linear algebra. Let $\mathbb{F}_2$ denote the field of two elements, 0 and 1, and $\mathbb{F}_2^n$ the vector space of dimension $n$ over $\mathbb{F}$. An assignment $\sigma$ can be thought of as an element of $\mathbb{F}_2^n$. Similarly, an XOR constraint $D$ can be seen as a linear constraint $a_1 x_1 + a_2 x_2 + \ldots + a_n x_n + b = 1$, where $a_i, b \in \{0,1\}$, $+$ denotes addition modulo 2 for $\mathbb{F}_2$, $a_i = 1$ iff $D$ has variable $i$, and $b = 1$ iff $D$ has the parity constant 1. In this setting, we can talk about linear transformations of $\mathbb{F}_2^n$ as well as linear independence of $\sigma, \sigma' \in \mathbb{F}_2^n$ (see standard texts for details). We will use two properties: every linear transformation maps the all-zeros vector to itself, and there exists a linear transformation that maps any $k$ linearly independent vectors to any other $k$ linearly independent vectors.

Consider the set $X$ of all XOR constraints over $V$. Since an XOR constraint is a subset of $V \cup \{1\}$, $|X| = 2^{n+1}$. Our method requires choosing XOR constraints from $X$ at random. Let $\mathbb{X}(n,q)$ denote the *probability distribution* over $X$ defined as follows: select each $v \in V$ independently at random with probability $q$ and include the constant 1 independently with probability $^1/_2$. This produces XORs of average length $nq$. In particular, note that every two complementary XOR constraints involving the same subset of $V$ (e.g., $c \oplus d$ and $c \oplus d \oplus 1$) are chosen with the same probability irrespective of $q$. Such complementary XOR constraints have the simple but useful property that any assignment $\sigma$ satisfies exactly one of them. Finally, when the distribution $\mathbb{X}(n,^1/_2)$ is used, every XOR constraint in $X$ is chosen with probability $2^{-(n+1)}$.

We will be interested in the random variables which are the sum of indicator random variables: $Y = \sum_\sigma Y_\sigma$. Linearity of expectation says that $\mathbb{E}[Y] = \sum_\sigma \mathbb{E}[Y_\sigma]$. When various $Y_\sigma$ are *pairwise independent*, i.e., knowing $Y_{\sigma_2}$ tells us nothing about $Y_{\sigma_1}$, even variance behaves linearly: $\text{Var}[Y] = \sum_\sigma \text{Var}[Y_\sigma]$. We will also need *conditional probabilities*. Here, for a random event $X$, linearity of conditional expectation says that $\mathbb{E}[Y \mid X] = \sum_\sigma \mathbb{E}[Y_\sigma \mid X]$. Let $X = Y_{\sigma_0}$. When various $Y_\sigma$ are *3-wise independent*, i.e., knowing $Y_{\sigma_2}$ and $Y_{\sigma_3}$ tells us nothing about $Y_{\sigma_1}$, even *conditional variance behaves linearly*: $\text{Var}[Y \mid Y_{\sigma_0}] = \sum_\sigma \text{Var}[Y_\sigma \mid Y_{\sigma_0}]$. This will be key to the analysis of our second algorithm.

# 3 Sampling using XOR constraints

In this section, we describe and analyze two randomized algorithms, XORSample and XORSample', for sampling solutions of a given Boolean formula $F$ near-uniformly using streamlining with random XOR constraints. Both algorithms are parameterized by two quantities: a positive integer $s$ and a real number $q \in (0,1)$, where $s$ is the number of XORs added to $F$ and $\mathbb{X}(n,q)$ is the distribution from which they are drawn. These parameters determine the degree of uniformity achieved by the algorithms, which we formalize as Theorems 1 and 2. The first algorithm, XORSample, uses a SAT solver as a subroutine on the randomly streamlined formula. It repeatedly performs the streamlining process until the resulting formula has a unique solution. When $s$ is chosen appropriately, it takes XORSample a small number of iterations (on average) to successfully produce a sample. The second algorithm, XORSample', is non-iterative. Here $s$ is chosen to be relatively small so that a moderate number of solutions survive. XORSample' then uses stronger subroutines, namely a SAT model counter and a model selector, to output one of the surviving solutions uniformly at random.

## 3.1 XOR-based sampling using SAT solvers: XORSample

Let $F$ be a formula over $n$ variables, and $q$ and $s$ be the parameters of XORSample. The algorithm works by adding to $F$, in each iteration, $s$ random XOR constraints $Q_s$ drawn independently from the distribution $\mathbb{X}(n,q)$. This generates a streamlined formula $F_s^q$ whose solutions (called the *surviving solutions*) are a subset of the solutions of $F$. If there is a unique surviving solution $\sigma$, XORSample outputs $\sigma$ and stops. Otherwise, it discards $Q_s$ and $F_s^q$, and iterates the process (rejection sampling). The check for uniqueness of $\sigma$ is done by adding the negation of $\sigma$ as a constraint to $F_s^q$ and testing whether the resulting formula is still satisfiable. See Algorithm 1 for a full description.

**Params**: $q \in (0,1)$, a positive integer $s$
**Input** : A CNF formula $F$
**Output**: A solution of $F$
**begin**
    $iterationSuccessful \leftarrow$ FALSE
    **while** $iterationSuccessful =$ FALSE **do**
        $Q_s \leftarrow \{s$ random constraints independently drawn from $\mathbb{X}(n,q)\}$
        $F_s^q \leftarrow F \cup Q_s$                     // Add $s$ random XOR constraints to $F$
        $result \leftarrow$ SATSolve$(F_s^q)$                // Solve using a SAT solver
        **if** $result =$ TRUE **then**
            $\sigma \leftarrow$ solution returned by SATSolve $(F_s^q)$
            $F' \leftarrow F_s^q \cup \{\bar{\sigma}\}$               // Remove $\sigma$ from the solution set
            $result' \leftarrow$ SATSolve$(F')$
            **if** $result' =$ FALSE **then**
                $iterationSuccessful =$ TRUE
                **return** $\sigma$          // Output $\sigma$; it is the unique solution of $F_s^q$
**end**

**Algorithm 1**: XORSample, sampling solutions with XORs using a SAT solver

We now analyze how uniform the samples produced by XORSample are. For the rest of this section, fix $q = \frac{1}{2}$. Let $F$ be satisfiable and have exactly $2^{s^*}$ solutions; $s^* \in [0,n]$. Ideally, we would like each solution $\sigma$ of $F$ to be sampled with probability $2^{-s^*}$. Let $p_{one,s}(\sigma)$ be the probability that XORSample outputs $\sigma$ in *one iteration*. This is typically much lower than $2^{-s^*}$, which is accounted for by rejection sampling. Nonetheless, we will show that when $s$ is larger than $s^*$, the variation in $p_{one,s}(\sigma)$ over different $\sigma$ is small. Let $p_s(\sigma)$ be the overall probability that XORSample outputs $\sigma$. This, we will show, is very close to $2^{-s^*}$, where "closeness" is formalized as being within a factor of $c(\alpha)$ which approaches 1 very fast. The proof closely follows the argument used by Valiant and

Vazirani [16] in their complexity theory work on unique satisfiability. However, we give a different, non-combinatorial argument for the pairwise independence property of XORs needed in the proof, relying on linear algebra. This approach is insightful and will come handy in Section 3.2. We describe the main idea below, deferring details to the full version of the paper.

**Lemma 1.** *Let* $\alpha > 0, c(\alpha) = 1 - 2^{-\alpha}$, *and* $s = s^* + \alpha$. *Then* $c(\alpha)2^{-s} < p_{one,s}(\sigma) \leq 2^{-s}$.

*Proof sketch.* We first prove the upper bound on $p_{one,s}(\sigma)$. Recall that for any two complementary XORs (e.g. $c \oplus d$ and $c \oplus d \oplus 1$), $\sigma$ satisfies exactly one XOR. Hence, the probability that $\sigma$ satisfies an XOR chosen randomly from the distribution $\mathbb{X}(n,q)$ is $^1/_2$. By independence of the $s$ XORs in $Q_s$ in XORSample, $\sigma$ survives with probability exactly $2^{-s}$, giving the desired upper bound on $p_{one,s}(\sigma)$.

For the lower bound, we resort to pairwise independence. Let $\sigma \neq \sigma'$ be two solutions of $F$. Let $D$ be an XOR chosen randomly from $\mathbb{X}(n,^1/_2)$. We use linear algebra arguments to show that the probability that $\sigma(D) = 1$ (i.e., $\sigma$ satisfies $D$) is independent of the probability that $\sigma'(D) = 1$. Recall the interpretation of variable assignments and XOR constraints in the vector space $\mathbb{F}_2^n$ (cf. Section 2). First suppose that $\sigma$ and $\sigma'$ are linearly dependent. In $\mathbb{F}_2^n$, this can happen only if exactly one of $\sigma$ and $\sigma'$ is the all-zeros vector. Suppose $\sigma = (0,0,\ldots,0)$ and $\sigma'$ is non-zero. Perform a linear transformation on $\mathbb{F}_2^n$ so that $\sigma' = (1,0,\ldots,0)$. Let $D$ be the constraint $a_1 x_1 + a_2 x_2 + \ldots + a_n x_n + b = 1$. Then, $\sigma'(D) = a_1 + b$ and $\sigma(D) = b$. Since $a_1$ is chosen uniformly from $\{0,1\}$ when $D$ is drawn from $\mathbb{X}(n,^1/_2)$, knowing $a_1 + b$ gives us no information about $b$, proving independence. A similar argument works when $\sigma$ is non-zero and $\sigma' = (0,0,\ldots,0)$, and also when $\sigma$ and $\sigma'$ are linearly independent to begin with. We skip the details.

This proves that $\sigma(D)$ and $\sigma'(D)$ are independent when $D$ is drawn from $\mathbb{X}(n,^1/_2)$. In particular, $\Pr[\sigma'(D) = 1 \mid \sigma(D) = 1] = ^1/_2$. This reasoning easily extends to $s$ XORs in $Q_s$ and we have that $\Pr[\sigma'(Q_s) = 1 \mid \sigma(Q_s) = 1] = 2^{-s}$. Now,

$$p_{one,s}(\sigma) = \Pr\left[\sigma(Q_s) = 1 \text{ and for all other solutions } \sigma' \text{ of } F, \sigma'(Q_s) = 0\right]$$
$$= \Pr[\sigma(Q_s) = 1] \cdot \left(1 - \Pr\left[\text{for some solution } \sigma' \neq \sigma, \sigma'(Q_s) = 1 \mid \sigma(Q_s) = 1\right]\right).$$

Evaluating this using the union bound and pairwise independence shows $p_{one,s}(\sigma) > c(\alpha)\,2^{-s}$. □

**Theorem 1.** *Let $F$ be a formula with $2^{s^*}$ solutions. Let $\alpha > 0, c(\alpha) = 1 - 2^{-\alpha}$, and $s = s^* + \alpha$. For any solution $\sigma$ of $F$, the probability $p_s(\sigma)$ with which XORSample with parameters $q = ^1/_2$ and $s$ outputs $\sigma$ satisfies*

$$c(\alpha)\,2^{-s^*} < p_s(\sigma) < \frac{1}{c(\alpha)}\,2^{-s^*} \qquad and \qquad \min_\sigma\{p_s(\sigma)\} > c(\alpha)\,\max_\sigma\{p_s(\sigma)\}.$$

*Further, the number of iterations needed to produce one sample has a geometric distribution with expectation between $2^\alpha$ and $2^\alpha/c(\alpha)$.*

*Proof.* Let $\hat{p}$ denote the probability that XORSample finds some unique solution in any single iteration. $p_{one,s}(\sigma)$, as before, is the probability that $\sigma$ is the unique surviving solution. $p_s(\sigma)$, the overall probability of sampling $\sigma$, is given by the infinite geometric series

$$p_s(\sigma) = p_{one,s}(\sigma) + (1 - \hat{p})p_{one,s}(\sigma) + (1 - \hat{p})^2 p_{one,s}(\sigma) + \ldots$$

which sums to $p_{one,s}(\sigma)/\hat{p}$. In particular, $p_s(\sigma)$ is proportional to $p_{one,s}(\sigma)$.

Lemma 1 says that for any two solutions $\sigma_1$ and $\sigma_2$ of $F$, $p_{one,s}(\sigma_1)$ and $p_{one,s}(\sigma_2)$ are strictly within a factor of $c(\alpha)$ of each other. By the above discussion, $p_s(\sigma_1)$ and $p_s(\sigma_2)$ must also be strictly within a factor of $c(\alpha)$ of each other, already proving the min vs. max part of the result. Further, $\sum_\sigma p_s(\sigma) = 1$ because of rejection sampling.

For the first part of the result, suppose for the sake of contradiction that $p_s(\sigma_0) \leq c(\alpha)2^{-s^*}$ for some $\sigma_0$, violating the claimed lower bound. By the above argument, $p_s(\sigma)$ is within a factor of $c(\alpha)$ of $p_s(\sigma_0)$ for every $\sigma$, and would therefore be at most $2^{-s^*}$. This would make $\sum_\sigma p_s(\sigma)$ strictly less than one, a contradiction. A similar argument proves the upper bound on $p_s(\sigma)$.

Finally, the number of iterations needed to find a unique solution (thereby successfully producing a sample) is a geometric random variable with success parameter $\hat{p} = \sum_\sigma p_{one,s}(\sigma)$, and has expected value $1/\hat{p}$. Using the bounds on $p_{one,s}(\sigma)$ from Lemma 1 and the fact that the unique survival of each of the $2^{s^*}$ solutions $\sigma$ are disjoint events, we have $\hat{p} \leq 2^{s^*}2^{-s} = 2^{-\alpha}$ and $\hat{p} > 2^{s^*}c(\alpha)2^{-s} = c(\alpha)2^{-\alpha}$. This proves the claimed bounds on the expected number of iterations, $1/\hat{p}$. □

## 3.2 XOR-based sampling using model counters and selectors: `XORSample'`

We now discuss our second parameterized algorithm, `XORSample'`, which also works by adding to $F$ $s$ random XORs $Q_s$ chosen independently from $\mathbb{X}(n,q)$. However, now the resulting streamlined formula $F_s^q$ is fed to an exact model counting subroutine to compute the number of surviving solutions, $mc$. If $mc > 0$, `XORSample'` *succeeds* and outputs the $i^{th}$ surviving solution using a model selector on $F_s^q$, where $i$ is chosen uniformly from $\{1, 2, \ldots, mc\}$. Note that `XORSample'`, in contrast to `XORSample`, is non-iterative. Also, the model counting and selecting subroutines it uses are more complex than SAT solvers; these work well in practice only because $F_s^q$ is highly streamlined.

**Params**: $q \in (0,1)$, a positive integer $s$
**Input**   : A CNF formula $F$
**Output**: A solution of $F$, or Failure
**begin**
  $Q_s \leftarrow \{s$ constraints randomly drawn from $\mathbb{X}(n, p)\}$
  $F_s^q \leftarrow F \cup Q_s$                           // Add $s$ random XOR constraints to $F$
  $mc \leftarrow$ SATModelCount($F_s^q$)            // Compute the exact model count of $F_s^q$
  **if** $mc \neq 0$ **then**
    $i \leftarrow$ a random number chosen uniformly from $\{1, 2, \ldots, mc\}$
    $\sigma \leftarrow$ SATFindSolution($F_s^q, i$)             // Compute the $i^{th}$ solution
    **return** $\sigma$                              // Sampled successfully!
  **else return** Failure
**end**

Algorithm 2: `XORSample'`, sampling with XORs using a model counter and selector

The sample-quality analysis of `XORSample'` requires somewhat more complex ideas than that of `XORSample`. Let $F$ have $2^{s^*}$ solutions as before. We again fix $q = 1/2$ and prove that if the parameter $s$ is sufficiently smaller than $s^*$, the sample-quality is provably good. The proof relies on the fact that XORs chosen randomly from $\mathbb{X}(n, 1/2)$ act *3-wise independently* on different solutions, i.e., knowing the value of an XOR constraint on two variable assignments does not tell us anything about its value on a third assignment. We state this as the following lemma, which can be proved by extending the linear algebra arguments we used in the proof of Lemma 1 (see the full version for details).

**Lemma 2 (3-wise independence).** *Let* $\sigma_1, \sigma_2$, *and* $\sigma_3$ *be three distinct assignments to n Boolean variables. Let D be an* XOR *constraint chosen at random from* $\mathbb{X}(n, 1/2)$. *Then for* $i \in \{0, 1\}$, $\Pr[\sigma_1(D) = i \mid \sigma_2(D), \sigma_3(D)] = \Pr[\sigma_1(D) = i]$.

Recall the discussion of expectation, variance, pairwise independence, and 3-wise independence in Section 2. In particular, when a number of random variables are 3-wise independent, the conditional variance of their sum (conditioned on one of these variables) equals the sum of their individual conditional variances. We use this to compute bounds on the sampling probability of `XORSample'`. The idea is to show that the number of solutions surviving, given that any fixed solution $\sigma$ survives, is independent of $\sigma$ in expectation and is highly likely to be very close to the expected value. As a result, the probability with which $\sigma$ is output, which is inversely proportional to the number of solutions surviving along with $\sigma$, will be very close to the uniform probability. Here "closeness" is one-sided and is measured as being within a factor of $c'(\alpha)$ which approaches 1 very quickly.

**Theorem 2.** *Let F be a formula with* $2^{s^*}$ *solutions. Let* $\alpha > 0$ *and* $s = s^* - \alpha$. *For any solution* $\sigma$ *of F, the probability* $p_s'(\sigma)$ *with which* `XORSample'` *with parameters* $q = 1/2$ *and s outputs* $\sigma$ *satisfies*

$$ p_s'(\sigma) \; > \; c'(\alpha) \, 2^{-s^*}, \qquad \text{where } \; c'(\alpha) = \frac{1 - 2^{-\alpha/3}}{(1 + 2^{-\alpha})(1 + 2^{-\alpha/3})}. $$

*Further,* `XORSample'` *succeeds with probability larger than* $c'(\alpha)$.

*Proof sketch.* See the full version for a detailed proof. We begin by setting up a framework for analyzing the number of surviving solutions after $s$ XORs $Q_s$ drawn from $\mathbb{X}(n, 1/2)$ are added to $F$. Let $Y_{\sigma'}$ be the indicator random variable which is 1 iff $\sigma'(Q_s) = 1$, i.e., $\sigma'$ survives $Q_s$. $\mathbb{E}[Y_{\sigma'}] = 2^{-s}$ and $\text{Var}[Y_{\sigma'}] \leq \mathbb{E}[Y_{\sigma'}] = 2^{-s}$. Further, a straightforward generalization of Lemma 2 from a single XOR constraint $D$ to $s$ independent XORs $Q_s$ implies that the random variables $Y_{\sigma'}$ are 3-wise independent.

The variable $mc$ (see Algorithm 2), which is the number of surviving solutions, equals $\sum_{\sigma'} Y_{\sigma'}$. Consider the distribution of $mc$ *conditioned* on the fact that $\sigma$ survives. Using pairwise independence, the corresponding conditional expectation can be shows to satisfy: $\mu = \mathbb{E}[mc \mid \sigma(Q_s) = 1] =$

$1 + (2^{s^*} - 1)2^{-s}$. More interesting, using 3-wise independence, the corresponding *conditional variation* can also be bounded: $\text{Var}[mc \mid \sigma(Q_s) = 1] < \mathbb{E}[mc \mid \sigma(Q_s) = 1]$.

Since $s = s^* - \alpha$, $2^\alpha < \mu < 1 + 2^\alpha$. We show that $mc$ conditioned on $\sigma(Q_s) = 1$ indeed lies very close to $\mu$. Let $\beta \geq 0$ be a parameter whose value we will fix later. By Chebychev's inequality,

$$\Pr\left[|mc - \mu| \geq \frac{\mu}{2^\beta} \mid \sigma(Q_s) = 1\right] \leq \frac{2^{2\beta}\,\text{Var}[mc \mid \sigma(Q_s) = 1]}{(\mathbb{E}[mc \mid \sigma(Q_s) = 1])^2} < \frac{2^{2\beta}}{\mathbb{E}[mc \mid \sigma(Q_s) = 1]} = \frac{2^{2\beta}}{\mu}$$

Therefore, conditioned on $\sigma(Q_s) = 1$, with probability more than $1 - 2^{2\beta}/\mu$, $mc$ lies between $(1 - 2^{-\beta})\mu$ and $(1 + 2^{-\beta})\mu$. Recall that $p'_s(\sigma)$ is the probability that XORSample' outputs $\sigma$.

$$p'_s(\sigma) = \Pr[\sigma(Q_s) = 1] \sum_{i=1}^{n} \Pr[mc = i \mid \sigma(Q_s) = 1]\frac{1}{i}$$

$$\geq 2^{-s}\,\Pr\left[mc \leq (1 + 2^{-\beta})\mu \mid \sigma(Q_s) = 1\right]\frac{1}{(1 + 2^{-\beta})\mu} \geq 2^{-s}\frac{1 - 2^{2\beta}/\mu}{(1 + 2^{-\beta})\mu}$$

Simplifying this expression and optimizing it by setting $\beta = \alpha/3$ gives the desired bound on $p'_s(\sigma)$. Lastly, the success probability of XORSample' is $\sum_\sigma p'_s(\sigma) > c'(\alpha)$. ☐

**Remark 1.** Theorems 1 and 2 show that both XORSample and XORSample' can be used to sample arbitrarily close to the uniform distribution when $q = 1/2$. For example, as the number of XORs used in XORSample is increased, $\alpha$ increases, the deviation $c(\alpha)$ from the truly uniform sampling probability $p^*$ approaches 0 exponentially fast, and we get progressively smaller error bands around $p^*$. However, for any fixed $\alpha$, these algorithms, somewhat counter-intuitively, do not always sample truly uniformly (see the full version). As a result, we expect to see a fluctuation around $p^*$, which, as we proved above, will be exponentially small in $\alpha$.

## 4  Empirical validation

To validate our XOR-sampling technique, we consider two kinds of formulas: a random 3-SAT instance generated near the SAT phase transition [18] and a structured instance derived from a logistics planning domain (data and code available from the authors). We used a complete model counter, Relsat [12], to find all solutions of our problem instances. Our random instance with 75 variables has a total of 48 satisfying assignments, and our logistics formula with 352 variables has 512 satisfying assignments. (We used formulas with a relatively small number of assignments in order to evaluate the quality of our sampling. Note that we need to draw many samples for each assignment.) We used XORSample with MiniSat [14] as the underlying SAT solver to generate samples from the set of solutions of each formula. Each sample took a fraction of a second to generate on a 4GHz processor. For comparison, we also ran the best alternative method for sampling from SAT problems, SampleSAT [19, 2], allowing it roughly the same cumulative runtime as XORSample.

Figure 1 depicts our results. In the left panel, we consider the random SAT instance, generating 200,000 samples total. In pure uniform sampling, in expectation we have $200,000/48 \approx 4,167$ samples for each solution. This level is indicated with the solid horizontal line. We see that the samples produced by XORSample all lie in a narrow band centered around this line. Contrast this with the results for SampleSAT: SampleSAT does sample quite uniformly from solutions that lie near each other in Hamming distance but different solution clusters are sampled with different frequencies. This SAT instance has two solution clusters: the first 32 solutions are sampled around 2,900 times each, i.e., not frequently enough, whereas the remaining 16 solutions are sampled too frequently, around 6,700 times each. (Although SampleSAT greatly improves on other sampling strategies for SAT, the split into disjoint sampling bands appears inherent in the approach.) The Kullback-Leibler (KL) divergence between the XORSample data and the uniform distribution is 0.002. For SampleSAT the KL-divergence from uniform is 0.085. It is clear that the XORSample approach leads to much more uniform sampling.

The right panel in Figure 1 gives the results for our structured logistics planning instance. (To improve the readability of the figure, we plot the sample frequency only for every fifth assignment.) In this case, the difference between XORSample and SampleSAT is even more dramatic. SampleSAT in fact only found 256 of the 512 solutions in a total of 100,000 samples. We also see that one of these solutions is sampled nearly 60,000 times, whereas many other solutions are sampled less than

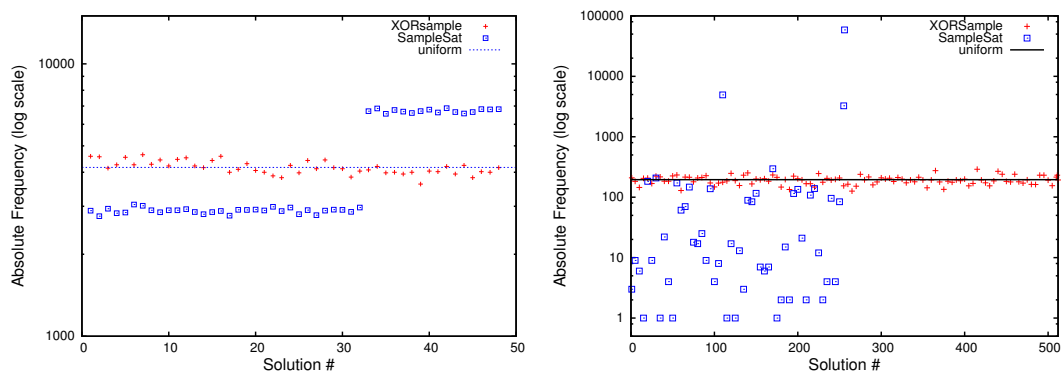

Figure 1: Results of `XORSample` and `SampleSAT` on a random 3-SAT instance, the left panel, and a logistics planning problem, the right panel. (See color figures in PDF.)

five times. The KL divergence from uniform is 4.16. (Technically the KL divergence is infinite, but we assigned a count of one to the non-sampled solutions.) The expected number of samples for each assignment is $100,000/512 \approx 195$. The figure also shows that the sample counts from `XORSample` all lie around this value; their KL divergence from uniform is 0.013.

These experiments show that `XORSample` is a promising practical technique (with theoretical guarantees) for obtaining near-uniform samples from intricate combinatorial spaces.

## Footnotes

*This work was supported by the Intelligent Information Systems Institute (IISI) at Cornell University (AFOSR grant F49620-01-1-0076) and DARPA (REAL grant FA8750-04-2-0216).

[1] Of course, we don't know the true value of $s$. In practice, we use a binary style search to obtain a rough estimate. As we will see, our algorithms work correctly even with over- and under-estimates for $s$.

[2] The practical feasibility of our approach exploits the fact that current SAT solvers are very effective in finding such truth assignments in many real-world domains.

[3] Note that there are certain classes of structured instances based on parity constraints that are designed to be hard for SAT solvers [17]. Our augmented problem instances appear to behave quite differently from these specially constructed instances because of the interaction between the constraints in the original instance and the added random parity constraints.

## References

[1] M. Richardson and P. Domingos. Markov logic networks. *Machine Learning*, 62(1-2):107–136, 2006.

[2] H. Poon and P. Domingos. Sound and efficient inference with probabilistic and deterministic dependencies. In *21th AAAI*, pages 458–463, Boston, MA, July 2006.

[3] N. Madras. Lectures on Monte Carlo methods. In *Field Institute Monographs*, vol. 16. Amer. Math. Soc., 2002.

[4] N. Metropolis, A. Rosenbluth, M. Rosenbluth, A. Teller, and E. Teller. Equations of state calculations by fast computing machines. *J. Chem. Phy.*, 21:1087–1092, 1953.

[5] S. Kirkpatrick, D. Gelatt Jr., and M. Vecchi. Optimization by simuleated annealing. *Science*, 220(4598): 671–680, 1983.

[6] D. Roth. On the hardness of approximate reasoning. *J. AI*, 82(1-2):273–302, 1996.

[7] M. L. Littman, S. M. Majercik, and T. Pitassi. Stochastic Boolean satisfiability. *J. Auto. Reas.*, 27(3): 251–296, 2001.

[8] J. D. Park. MAP complexity results and approximation methods. In *18th UAI*, pages 388–396, Edmonton, Canada, August 2002.

[9] A. Darwiche. The quest for efficient probabilistic inference, July 2005. Invited Talk, IJCAI-05.

[10] T. Sang, P. Beame, and H. A. Kautz. Performing Bayesian inference by weighted model counting. In *20th AAAI*, pages 475–482, Pittsburgh, PA, July 2005.

[11] F. Bacchus, S. Dalmao, and T. Pitassi. Algorithms and complexity results for #SAT and Bayesian inference. In *44nd FOCS*, pages 340–351, Cambridge, MA, October 2003.

[12] R. J. Bayardo Jr. and R. C. Schrag. Using CSP look-back techniques to solve real-world SAT instances. In *14th AAAI*, pages 203–208, Providence, RI, July 1997.

[13] L. Zhang, C. F. Madigan, M. H. Moskewicz, and S. Malik. Efficient conflict driven learning in a Boolean satisfiability solver. In *ICCAD*, pages 279–285, San Jose, CA, November 2001.

[14] N. Eén and N. Sörensson. MiniSat: A SAT solver with conflict-clause minimization. In *8th SAT*, St. Andrews, U.K., June 2005. Poster.

[15] C. P. Gomes, A. Sabharwal, and B. Selman. Model counting: A new strategy for obtaining good bounds. In *21th AAAI*, pages 54–61, Boston, MA, July 2006.

[16] L. G. Valiant and V. V. Vazirani. NP is as easy as detecting unique solutions. *Theoretical Comput. Sci.*, 47(3):85–93, 1986.

[17] J. M. Crawford, M. J. Kearns, and R. E. Schapire. The minimal disagreement parity problem as a hard satisfiability problem. Technical report, AT&T Bell Labs., 1994.

[18] D. Achlioptas, A. Naor, and Y. Peres. Rigorous location of phase transitions in hard optimization problems. *Nature*, 435:759–764, 2005.

[19] W. Wei, J. Erenrich, and B. Selman. Towards efficient sampling: Exploiting random walk strategies. In *19th AAAI*, pages 670–676, San Jose, CA, July 2004.
